# Selecting Observations against Adversarial Objectives

**Andreas Krause**
SCS, CMU

**H. Brendan McMahan**
Google, Inc.

**Carlos Guestrin**
SCS, CMU

**Anupam Gupta**
SCS, CMU

## Abstract

In many applications, one has to actively select among a set of expensive observations before making an informed decision. Often, we want to select observations which perform well when evaluated with an objective function chosen by an adversary. Examples include minimizing the maximum posterior variance in Gaussian Process regression, robust experimental design, and sensor placement for outbreak detection. In this paper, we present the *Submodular Saturation* algorithm, a simple and efficient algorithm with strong theoretical approximation guarantees for the case where the possible objective functions exhibit *submodularity*, an intuitive diminishing returns property. Moreover, we prove that better approximation algorithms do not exist unless NP-complete problems admit efficient algorithms. We evaluate our algorithm on several real-world problems. For Gaussian Process regression, our algorithm compares favorably with state-of-the-art heuristics described in the geostatistics literature, while being simpler, faster and providing theoretical guarantees. For robust experimental design, our algorithm performs favorably compared to SDP-based algorithms.

## 1 Introduction

In tasks such as sensor placement for environmental temperature monitoring or experimental design, one has to select among a large set of possible, but expensive, observations. Often, there are several different objective functions which we want to simultaneously optimize. For example, in the environmental monitoring problem, we want to minimize the marginal posterior variance of our temperature estimate at all locations simultaneously. In experimental design, we often have uncertainty about the model parameters, and we want our experiments to be informative no matter what the true parameters of the model are. These problems can be interpreted as a game: We select a set of observations (sensor locations, experiments), and an adversary selects an objective function (location to evaluate predictive variance, model parameters etc.) to test us on. Often, the individual objective functions (e.g., the marginal variance at one location, or the information gain for a fixed set of parameters [1, 2]) satisfy *submodularity*, an intuitive diminishing returns property: Adding a new observation helps less if we have already made many observations, and more if we have made few observation thus far. While NP-hard, the problem of selecting an optimal set of $k$ observations maximizing a single submodular objective can be approximately solved using a simple greedy forward-selection algorithm, which is guaranteed to perform near-optimally [3]. However, as we show, this simple *myopic* algorithm performs arbitrarily badly in the case of an adversarially chosen objective.

In this paper, we address this problem. In particular: (1) We present SATURATE, an efficient algorithm for settings where an adversarially-chosen submodular objective function must be optimized. Our algorithm guarantees solutions which are at least as informative as the optimal solution, at only a slightly higher cost. (2) We prove that our approximation guarantee is best possible and cannot be improved unless NP-complete problems admit efficient algorithms. (3) We extensively evaluate our algorithm on several real-world tasks, including minimizing the maximum posterior variance in Gaussian Process regression, finding experiment designs which are robust with respect to parameter uncertainty, and sensor placement for outbreak detection.

## 2 The adversarial observation selection problem

**Observation selection with a single submodular objective.** Observation selection problems can often be modeled using set functions: We have a finite set $\mathcal{V}$ of observations to choose from, and

a utility function $F$ which assigns a real number $F(\mathcal{A})$ to each $\mathcal{A} \subseteq \mathcal{V}$, quantifying its informativeness. In many settings, such as the ones described above, the utility $F$ exhibits the property of *submodularity*: adding an observation helps more, the fewer observations made so far [2]. Formally, $F$ is submodular [3] if, for all $\mathcal{A} \subseteq \mathcal{B} \subseteq \mathcal{V}$ and $s \in \mathcal{V} \setminus \mathcal{B}$, it holds that $F(\mathcal{A} \cup \{s\}) - F(\mathcal{A}) \geq F(\mathcal{B} \cup \{s\}) - F(\mathcal{B})$; $F$ is *monotonic* if for all $\mathcal{A} \subseteq \mathcal{B} \subseteq \mathcal{V}$ it holds that $F(\mathcal{A}) \leq F(\mathcal{B})$, and $F$ is *normalized* if $F(\emptyset) = 0$. Hence, many observation selection problems can be formalized as

$$\max_{\mathcal{A} \subseteq \mathcal{V}} F(\mathcal{A}), \quad \text{subject to} \quad |\mathcal{A}| \leq k, \tag{2.1}$$

where $F$ is normalized, monotonic and submodular, and $k$ is a bound on the number of observations we can make. Since solving the problem (2.1) is generally NP-hard [4], in practice heuristics are often used. One such heuristic is the *greedy algorithm*. This algorithm starts with the empty set, and iteratively adds the element $s^* = \operatorname{argmax}_{s \in \mathcal{V} \setminus \mathcal{A}} F(\mathcal{A} \cup \{s\})$, until $k$ elements have been selected. Perhaps surprisingly, a fundamental result by Nemhauser et. al. [3] states that for submodular functions, the greedy algorithm achieves a constant factor approximation: The set $\mathcal{A}_G$ obtained by the greedy algorithm achieves at least a constant fraction $(1 - 1/e)$ of the objective value obtained by the optimal solution, i.e., $F(\mathcal{A}_G) \geq (1 - 1/e) \max_{|\mathcal{A}| \leq k} F(\mathcal{A})$. Moreover, no polynomial time algorithm can provide a better approximation guarantee unless P = NP [4].

**Observation selection with adversarial objectives.** In many applications (such as those discussed below), one wants to simultaneously optimize *multiple* objectives. Here, we are given a *collection* of monotonic submodular functions $F_1, \ldots, F_m$, and we want to solve

$$\max_{\mathcal{A} \subseteq \mathcal{V}} \min_i F_i(\mathcal{A}), \quad \text{subject to} \quad |\mathcal{A}| \leq k. \tag{2.2}$$

Problem (2.2) can be considered a *game*: First, we (the max-player) select a set of observations $\mathcal{A}$, and then our opponent (the min-player) selects a criterion $F_i$ to test us on. Our goal is to select a set $\mathcal{A}$ of observations which performs well against an opponent who chooses the worst possible $F_i$ knowing our choice $\mathcal{A}$. Thereby, we try to find a pure equilibrium to a sequential game on a matrix, with one row per $\mathcal{A}$, and one column per $F_i$. Note, that even if the $F_i$ are all submodular, $G(\mathcal{A}) = \min_i F_i(\mathcal{A})$ is *not* submodular. In fact, we show below that, in this setting, the simple greedy algorithm (which performs near-optimally in the single-criterion setting) can perform arbitrarily badly.

**Examples of adversarial observation selection problems.** We consider three instances of adversarial selection problems. Sec. 4 provides more details and experimental results for these domains. Several more examples are presented in the longer version of this paper [5].

*Minimizing the maximum Kriging variance.* Consider a Gaussian Process (GP) [6] $\mathcal{X}_{\mathcal{V}}$ defined over a finite set of locations (indices) $\mathcal{V}$. Hereby, $\mathcal{X}_{\mathcal{V}}$ is a set of random variables, one variable $\mathcal{X}_s$ for each location $s \in \mathcal{V}$. Given a set of locations $\mathcal{A} \subseteq \mathcal{V}$ which we observe, we can compute the predictive distribution $P(\mathcal{X}_{\mathcal{V} \setminus \mathcal{A}} \mid \mathcal{X}_{\mathcal{A}} = \mathbf{x}_{\mathcal{A}})$, i.e., the distribution of the variables $\mathcal{X}_{\mathcal{V} \setminus \mathcal{A}}$ at the unobserved locations $\mathcal{V} \setminus \mathcal{A}$, conditioned on the measurements at the selected locations, $\mathcal{X}_{\mathcal{A}} = \mathbf{x}_{\mathcal{A}}$. Let $\sigma^2_{s|\mathcal{A}}$ be the residual variance after making observations at $\mathcal{A}$. Let $\Sigma_{\mathcal{A}\mathcal{A}}$ be the covariance matrix of the measurements at the chosen locations $\mathcal{A}$, and $\Sigma_{s\mathcal{A}}$ be the vector of cross-covariances between the measurements at $s$ and $\mathcal{A}$. Then, the variance $\sigma^2_{s|\mathcal{A}} = \sigma^2_s - \Sigma_{s\mathcal{A}} \Sigma^{-1}_{\mathcal{A}\mathcal{A}} \Sigma_{\mathcal{A}s}$ depends only on the set $\mathcal{A}$, and *not* on the observed values $\mathbf{x}_{\mathcal{A}}$. Assume that the a priori variance $\sigma^2_s$ is constant for all locations $s$ (in Sec. 3, we show our approach generalizes to non-constant marginal variances). We want to select locations $\mathcal{A}$ such that the maximum marginal variance is as small as possible. Equivalently, we can define the *variance reduction* $F_s(\mathcal{A}) = \sigma^2_s - \sigma^2_{s|\mathcal{A}}$, and desire that the minimum variance reduction over all locations $s$ is as large as possible. Das and Kempe [1] show that, in many practical cases, the variance reduction $F_s$ is a monotonic submodular function.

*Robust experimental designs.* Another application is experimental design under nonlinear dynamics [7]. The goal is to estimate a set of parameters $\theta$ of a nonlinear function $y = f(\mathbf{x}, \theta) + w$, by providing a set of experimental stimuli $\mathbf{x}$, and measuring the (noisy) response $y$. In many cases, experimental design for linear models (where $y = A(\mathbf{x})^T \theta + w$) with Gaussian noise $w$ can be efficiently solved [8]. In the nonlinear case, the common approach is to *linearize* $f$ around an initial parameter estimate $\theta_0$, i.e., $y = f(\mathbf{x}, \theta_0) + V(\mathbf{x})(\theta - \theta_0) + w$, where $V(\mathbf{x})$ is the Jacobian of $f$ with respect to the parameters $\theta$, evaluated at $\theta_0$. In [7], it was shown that the efficiency of the design can be very sensitive with respect to the initial parameter estimates $\theta_0$. Consequently, they develop an efficient semi-definite program (SDP) for E-optimal design (i.e., the goal is to minimize the maximum eigenvalue of the error covariance) which is robust against perturbations of the Jacobian

$V$. However, it might be more natural to directly consider robustness with respect to perturbation of the initial parameter estimates $\theta_0$, around which the linearization is performed. We show how to find (Bayesian A-optimal) designs which are robust against uncertainty in these parameter estimates. In this setting, the objectives $F_{\theta_0}(\mathcal{A})$ are the reductions of the trace of the parameter covariance, $F_{\theta_0}(\mathcal{A}) = \mathrm{tr}(\Sigma_\theta^{(\theta_0)}) - \mathrm{tr}(\Sigma_{\theta|\mathcal{A}}^{(\theta_0)})$, where $\Sigma^{(\theta_0)}$ is the joint covariance of observations and parameters after linearization around $\theta_0$; thus, $F_{\theta_0}$ is the sum of marginal parameter variance reductions, which are individually monotonic and (often) submodular [1], and so $F_{\theta_0}$ is monotonic and submodular as well. Hence, in order to find a robust design, we maximize the minimum variance reduction, where the minimum is taken over (a discretization into a finite subset of) all initial parameter values $\theta_0$.

*Sensor placement for outbreak detection.* Another class of examples are outbreak detection problems on graphs, such as contamination detection in water distribution networks [9]. Here, we are given a graph $\mathcal{G} = (\mathcal{V}, \mathcal{E})$, and a phenomenon spreading dynamically over the graph. We define a set of *intrusion scenarios* $\mathcal{I}$; each scenario $i \in \mathcal{I}$ models an outbreak (e.g., spreading of contamination) starting from a given node $s \in \mathcal{V}$ in the network. By placing sensors at a set of locations $\mathcal{A} \subseteq \mathcal{V}$, we can detect such an outbreak, and incur a utility $F_i(\mathcal{A})$ (e.g., reduction in detection time or population affected). In [9], it was shown that these utilities $F_i$ are monotonic and submodular for a large class of utility functions. In the adversarial setting, the adversary observes our sensor placement $\mathcal{A}$, and then decides on an intrusion $i$ for which our utility $F_i(\mathcal{A})$ is as small as possible. Hence, our goal is to find a placement $\mathcal{A}$ which performs well against such an adversarial opponent.

**Hardness of the adversarial observation selection problem.** Given the near-optimal performance of the greedy algorithm for the single-objective problem, a natural question is if the performance guarantee generalizes to the more complex adversarial setting. Unfortunately, this is far from true. Consider the case with two submodular functions, $F_1$ and $F_2$, where the set of observations is $\mathcal{V} = \{s_1, s_2, t_1, t_2\}$. We set $F_1(\emptyset) = F_2(\emptyset) = 0$, and define $F_1(\mathcal{A}) = 1$ if $s_1 \in \mathcal{A}$, otherwise $\varepsilon$ times the number of $t_i$ contained in $\mathcal{A}$. Similarly, if $s_2 \in \mathcal{A}$, we set $F_2(\mathcal{A}) = 1$, otherwise $\varepsilon$ times the number of $t_i$ contained in $\mathcal{A}$. Both $F_1$ and $F_2$ are submodular and monotonic. Optimizing for a set of 2 elements, the greedy algorithm maximizing $G(\mathcal{A}) = \min\{F_1(\mathcal{A}), F_2(\mathcal{A})\}$ would choose the set $\{t_1, t_2\}$, since such choice increases $G$ by $2\varepsilon$, whereas adding $s_i$ would not increase the score. However, the optimal solution with $k = 2$ is $\{s_1, s_2\}$, with a score of 1. Hence, as $\varepsilon \to 0$, the greedy algorithm performs arbitrarily worse than the optimal solution. Our next hope would be to obtain a different good approximation algorithm. However, we can show that most likely this is not possible:

**Theorem 1.** *Unless* $\mathrm{P} = \mathrm{NP}$, *there cannot exist any polynomial time approximation algorithm for Problem* (2.2). *More precisely: Let $n$ be the size of the problem instance, and $\gamma(\cdot) > 0$ be any positive function of $n$. If there exists a polynomial-time algorithm which is guaranteed to find a set $\mathcal{A}'$ of size $k$ such that* $\min_i F_i(\mathcal{A}') \geq \gamma(n) \max_{|\mathcal{A}| \leq k} \min_i F_i(\mathcal{A})$, *then* $\mathrm{P} = \mathrm{NP}$.

Thus, unless $\mathrm{P} = \mathrm{NP}$, there cannot exist any algorithm which is guaranteed to provide, e.g., even an exponentially small fraction ($\gamma(n) = 2^{-n}$) of the optimal solution. All proofs can be found in [5].

## 3 The Submodular Saturation Algorithm

Since Theorem 1 rules out *any* approximation algorithm which respects the constraint $k$ on the size of the set $\mathcal{A}$, our only hope for non-trivial guarantees requires us to relax this constraint. We now present an algorithm that finds a set of observations which perform at least as well as the optimal set, but at slightly increased cost; moreover, we show that no efficient algorithms can provide better guarantees (under reasonable complexity-theoretic assumptions). For now we assume all $F_i$ take only integral values; this assumption is relaxed later. The key idea is to consider the following alternative formulation:

$$\max_{c, \mathcal{A}} c, \qquad \text{subject to} \quad c \leq F_i(\mathcal{A}) \text{ for } 1 \leq i \leq m \text{ and } |\mathcal{A}| \leq \alpha k. \qquad (3.1)$$

We want a set $\mathcal{A}$ of size at most $\alpha k$, such that $F_i(\mathcal{A}) \geq c$ for all $i$, and $c$ is as large as possible. Here $\alpha \geq 1$ is a parameter relaxing the constraint on $|\mathcal{A}|$: if $\alpha = 1$, we recover the original problem (2.2). We solve program (3.1) as follows: For each value $c$, we find the cheapest set $\mathcal{A}$ with $F_i(\mathcal{A}) \geq c$ for all $i$. If this cheapest set has at most $\alpha k$ elements, then $c$ is feasible. A binary search on $c$ allows us to find the optimal solution with the maximum feasible $c$. We first show how to *approximately* solve Equation (3.1) for a fixed $c$. For $c > 0$ define $\widehat{F}_{i,c}(\mathcal{A}) = \min\{F_i(\mathcal{A}), c\}$, the original function $F_i$ truncated at score level $c$; these $\widehat{F}_{i,c}$ functions are also submodular [10].

```
GPC ($\overline{F}_c$, c)
$\mathcal{A} \leftarrow \emptyset$;
while $\overline{F}_c(\mathcal{A}) < c$ do
    foreach $s \in \mathcal{V} \setminus \mathcal{A}$ do  $\delta_s \leftarrow \overline{F}_c(\mathcal{A} \cup \{s\}) - \overline{F}_c(\mathcal{A})$;
    $\mathcal{A} \leftarrow \mathcal{A} \cup \{\mathrm{argmax}_s \, \delta_s\}$;
```
**Algorithm 1**: The greedy submodular partial cover (GPC) algorithm.

```
SATURATE ($F_1, \ldots, F_m, k, \alpha$)
$c_{\min} \leftarrow 0$; $c_{\max} \leftarrow \min_i F_i(\mathcal{V})$; $\mathcal{A}_{best} \leftarrow \emptyset$;
while ($c_{\max} - c_{\min}$) $\geq \frac{1}{m}$ do
    $c \leftarrow (c_{\min} + c_{\max})/2$; $\forall \mathcal{A}$ define $\overline{F}_c(\mathcal{A}) \leftarrow \frac{1}{m} \sum_i \min\{F_i(\mathcal{A}), c\}$; $\mathcal{A} \leftarrow GPC(\overline{F}_c, c)$;
    if $|\mathcal{A}| > \alpha k$ then  $c_{\max} \leftarrow c$; else  $c_{\min} \leftarrow c$; $\mathcal{A}_{best} = \mathcal{A}$ ;
```
**Algorithm 2**: The Submodular Saturation algorithm.

Let $\overline{F}_c(\mathcal{A}) = \frac{1}{m} \sum_i \widehat{F}_{i,c}(\mathcal{A})$ be their average value; submodular functions are closed under convex combinations, so $\overline{F}_c$ is submodular and monotonic. Furthermore, $F_i(\mathcal{A}) \geq c$ for all $1 \leq i \leq m$ if and only if $\overline{F}_c(\mathcal{A}) = c$. Hence, in order to determine whether some $c$ is feasible, we solve a *submodular covering problem*:

$$\mathcal{A}_c = \mathrm{argmin}_{\mathcal{A} \subseteq \mathcal{V}} |\mathcal{A}|, \quad \text{such that} \quad \overline{F}_c(\mathcal{A}) = c. \tag{3.2}$$

Such problems are NP-hard in general [4], but in [11] it is shown that the greedy algorithm (*c.f.*, Algorithm 1) achieves near-optimal performance on this problem. Using this result, we find:

**Lemma 2.** *Given monotonic submodular functions $F_1, \ldots, F_m$ and a (feasible) constant $c$, Algorithm 1 (with input $\overline{F}_c$) finds a set $\mathcal{A}_G$ such that $F_i(\mathcal{A}_G) \geq c$ for all $i$, and $|\mathcal{A}_G| \leq \alpha |\mathcal{A}^*|$, where $\mathcal{A}^*$ is the optimal solution, and $\alpha = 1 + \log\left(\max_{s \in \mathcal{V}} \sum_i F_i(s)\right) \geq 1 + \log\left(m \max_{s \in \mathcal{V}} \overline{F}_c(s)\right)$.*[1]

We can compute this approximation guarantee $\alpha$ for any given instance of the adversarial observation selection problem. Hence, if for a given value of $c$ the greedy algorithm returns a set of size greater than $\alpha k$, there cannot exist a solution $\mathcal{A}'$ with $|\mathcal{A}'| \leq k$ with $F_i(\mathcal{A}') \geq c$ for all $i$; thus, the optimal solution to the adversarial observation selection problem must be less than $c$. We can use this argument to conduct a binary search to find the optimal value of $c$. We call Algorithm 2, which formalizes this procedure, the *submodular saturation algorithm* (SATURATE), as the algorithm considers the truncated objectives $\widehat{F}_{i,c}$, and chooses sets which *saturate* all these objectives. Theorem 3 (given below) states that SATURATE is guaranteed to find a set which achieves adversarial score $\min_i F_i$ at least as high as the optimal solution, if we allow the set to be logarithmically larger than the optimal solution.

**Theorem 3.** *For any integer $k$, SATURATE finds a solution $\mathcal{A}_S$ such that $\min_i F_i(\mathcal{A}_S) \geq \max_{|\mathcal{A}| \leq k} \min_i F_i(\mathcal{A})$ and $|\mathcal{A}_S| \leq \alpha k$, for $\alpha = 1 + \log\left(\max_{s \in \mathcal{V}} \sum_i F_i(s)\right)$. The total number of submodular function evaluations is $\mathcal{O}\left(|\mathcal{V}|^2 m \log(\sum_i F_i(\mathcal{V}))\right)$.*

Note, that the algorithm still makes sense for any value of $\alpha$. However, if $\alpha < 1 + \log\left(\max_{s \in \mathcal{V}} \sum_i F_i(s)\right)$, the guarantee of Theorem 3 does not hold. If we had an exact algorithm for submodular coverage, $\alpha = 1$ would be the correct choice. Since the greedy algorithm solves submodular coverage very effectively, in our experiments, we call SATURATE with $\alpha = 1$, which empirically performs very well. The worst-case running time guarantee is quite pessimistic, and in practice the algorithm is much faster: Using a priority queue and lazy evaluations, Algorithm 1 can be sped up drastically (*c.f.*, [12] for details). Furthermore, in practical implementations, one would stop GPC once $\alpha k + 1$ elements have been selected, which already proves that the optimal solution with $k$ elements cannot achieve score $c$. Also, Algorithm 2 can be terminated once $c_{\max} - c_{\min}$ is sufficiently small; in our experiments, 10-15 iterations usually sufficed.

One might ask, whether the guarantee on the size of the set, $\alpha$, can be improved. Unfortunately, this is not likely, as the following Theorem shows:

**Theorem 4.** *If there were a polynomial time algorithm which, for any integer $k$, is guaranteed to find a solution $\mathcal{A}_S$ such that $\min_i F_i(\mathcal{A}_S) \geq \max_{|\mathcal{A}| \leq k} \min_i F_i(\mathcal{A})$ and $|\mathcal{A}_S| \leq \beta k$, where $\beta \leq (1 - \varepsilon)(1 + \log \max_{s \in \mathcal{V}} \sum_i F_i(s))$ for some fixed $\varepsilon > 0$, then $\mathrm{NP} \subseteq \mathrm{DTIME}(n^{\log \log n})$.*

Hereby, $\mathrm{DTIME}(n^{\log \log n})$ is a class of deterministic, slightly superpolynomial (but sub-exponential) algorithms [4]; the inclusion $\mathrm{NP} \subseteq \mathrm{DTIME}(n^{\log \log n})$ is considered unlikely [4].

**Extensions.** We now show how the assumptions made in our presentation above can be relaxed.

*Non-integral objectives.* Most objective functions $F_i$ in the observation selection setting are not integral (e.g., marginal variances of GPs). If they take rational numbers, we can scale the objectives by multiplying by their common denominator. If we allow small additive error, we can approximate their values by their leading digits. An analysis similar to the one presented in [2] can be used to bound the effect of this approximation on the theoretical guarantees obtained by the algorithm.

*Non-constant thresholds.* Consider the example of Minimax Kriging Designs for GP regression. Here, the $F_i(\mathcal{A}) = \sigma_i^2 - \sigma_{i|\mathcal{A}}^2$ denote the variance reductions at location $i$. However, rather than guaranteeing that $F_i(\mathcal{A}) \geq c$ for all $i$ (which, in this example, means that the *minimum variance reduction* is $c$), we want to guarantee that $\sigma_{i|\mathcal{A}}^2 \leq c$ for all $i$. We can easily adapt our approach to handle this case: Instead of defining $\widehat{F}_{i,c}(\mathcal{A}) = \min\{F_i(\mathcal{A}), c\}$, we define $\widehat{F}_{i,c}(\mathcal{A}) = \min\{F_i(\mathcal{A}), \sigma_i^2 - c\}$, and then again perform binary search over $c$, but searching for the smallest $c$ instead. The algorithm, using objectives modified in this way, will bear the same approximation guarantees.

*Non-uniform observation costs.* We can extend SATURATE to the setting where different observations have different costs. Suppose a cost function $g : \mathcal{V} \to \mathbb{R}^+$ assigns each element $s \in \mathcal{V}$ a positive cost $g(s)$; the cost of a set of observations is then $g(\mathcal{A}) = \sum_{s \in \mathcal{A}} g(s)$. The problem is to find $\mathcal{A}^* = \max_{\mathcal{A} \subset \mathcal{V}} \min_i F_i(\mathcal{A})$ subject to $g(\mathcal{A}) \leq B$, where $B > 0$ is a *budget* we can spend on making observations. In this case, we use the rule $\delta_s \leftarrow \left(\overline{F}_c(\mathcal{A} \cup \{s\}) - \overline{F}_c(\mathcal{A})\right) / g(s)$ in Algorithm 1. For this modified algorithm, Theorem 3 still holds, with $|\mathcal{A}|$ replaced by $g(\mathcal{A})$ and $k$ replaced by $B$.

# 4 Experimental Results

**Minimax Kriging.** We use SATURATE to select observations in a GP to minimize the maximum posterior variance. We consider Precipitation data from the Pacific Northwest of the United States [13]. We discretize the space into 167 locations. In order to estimate variance reduction, we consider the empirical covariance of 50 years of data, which we preprocessed as described in [2].

In the geostatistics literature, the predominant choice of optimization algorithms are carefully tuned local search procedures, prominently simulated annealing (*c.f.*, [14, 15]). We compare our SATURATE algorithm against a state-of-the-art implementation of such a simulated annealing (SA) algorithm, first proposed by [14]. We use an optimized implementation described recently by [15]. This algorithm has 7 parameters which need to be tuned, describing the annealing schedule, distribution of iterations among several inner loops, etc. We use the parameter settings as reported by [15], and report the best result of the algorithm among 10 random trials. In order to compare observation sets of the same size, we called SATURATE with $\alpha = 1$.

Fig. 1(a) compares simulated annealing, SATURATE, and the greedy algorithm which greedily selects elements which decrease the maximum variance the most. We also used SATURATE to initialize the simulated annealing algorithm (using only a single run of simulated annealing, as opposed to 10 random trials). SATURATE obtains placements which are drastically better than the placements obtained by the greedy algorithm. Furthermore, the performance is very close to the performance of the simulated annealing algorithm. When selecting 30 and more sensors, SATURATE strictly outperforms the simulated annealing algorithm. Furthermore, as Fig. 1(b) shows, SATURATE is significantly faster than simulated annealing, by factors of 5-10 for larger problems. When using SATURATE in order to initialize the simulated annealing algorithm, the resulting performance almost always resulted in the best solutions we were able to find, while still executing faster than simulated annealing with 10 random restarts as proposed by [15]. These results indicate that SATURATE compares favorably to state-of-the-art local search heuristics, while being faster, requiring no parameters to tune, and providing theoretical approximation guarantees.

Optimizing for the maximum variance could potentially be considered too pessimistic. Hence we compared placements obtained by SATURATE, minimizing the maximum marginal posterior variance, with placements obtained by the greedy algorithm, where we minimize the *average* marginal variance. Note, that, whereas the reduction of the maximum variance is non-submodular, the *average* variance reduction is (often) submodular [1], and hence the greedy algorithm can be expected to provide near-optimal placements. Fig. 1(c) presents the maximum and average marginal variances for both algorithms. Our results show that if we optimize for the maximum variance we still achieve comparable average variance. If we optimize for average variance however, the

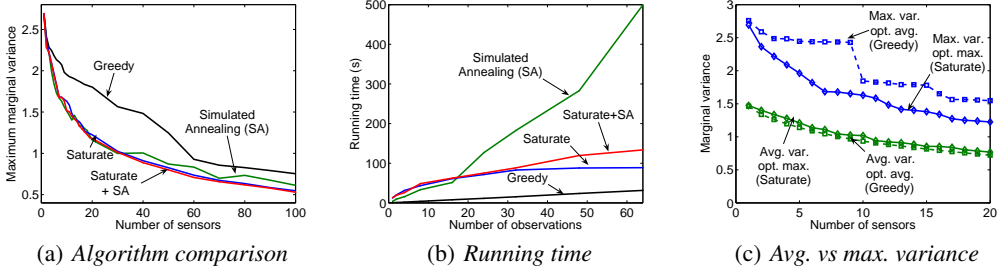

(a) *Algorithm comparison*  (b) *Running time*  (c) *Avg. vs max. variance*

Figure 1: (a) SATURATE, greedy and SA on the precipitation data. SATURATE performs comparably with the fine-tuned SA algorithm, and outperforms it for larger placements. (b) Running times for the same experiment. (c) Optimizing for the maximum variance (using SATURATE) leads to low average variance, but optimizing for average variance (using greedy) does not lead to low maximum variance.

maximum posterior variance remains much higher. In the longer version of this paper [5], we present results on two more real data sets, which are qualitatively similar to those discussed here.

**Robust Experimental Design.** We consider the robust design of experiments for the Michaelis-Menten mass-action kinetics model, as discussed in [7]. The goal is least-square parameter estimation for a function $y = f(x, \theta)$, where $x$ is the chosen experimental stimulus (the initial substrate concentration $S_0$), and $\theta = (\theta_1, \theta_2)$ are two parameters as described in [7]. The stimulus $x$ is chosen from a menu of six options, $x \in \{1/8, 1, 2, 4, 8, 16\}$, each of which can be repeatedly chosen. The goal is to produce a fractional design $\mathbf{w} = (w_1, \ldots, w_6)$, where each component $w_i$ measures the relative frequency according to which the stimulus $x_i$ is chosen. Since $f$ is nonlinear, $f$ is linearized around an initial parameter estimate $\theta_0 = (\theta_{01}, \theta_{02})$, and approximated by its Jacobian $V_{\theta_0}$. Classical experimental design considers the error covariance of the least squares estimate $\hat{\theta}$, $\text{Cov}(\hat{\theta} \mid \theta_0, \mathbf{w}) = \sigma^2 (V_{\theta_0}^T W V_{\theta_0})^{-1}$, where $W = \text{diag}(\mathbf{w})$, and aims to find designs $\mathbf{w}$ which minimize this error covariance. E-optimality, the criterion adopted by [7], measures smallness in terms of the maximum eigenvalue of the error covariance matrix. The optimal $\mathbf{w}$ can be found using Semidefinite Programming (SDP) [8].

The estimate $\text{Cov}(\hat{\theta} \mid \theta_0, \mathbf{w})$ depends on the initial parameter estimate $\theta_0$, where linearization is performed. However, since the goal is parameter estimation, a "certain circularity is involved" [7]. To avoid this problem, [7] find a design $\mathbf{w}_\rho(\theta_0)$ by solving a robust SDP which minimizes the error size, subject to a worst-case (adversarially-chosen) perturbation $\Delta$ on the Jacobian $V_{\theta_0}$; the robustness parameter $\rho$ bounds the spectral norm of $\Delta$. As evaluation criterion, [7] define a notion of *efficiency*, which is the error size of the optimal design with correct initial parameter estimate, divided by the error when using a robust design obtained at the wrong initial parameter estimates, i.e.,

$$\text{efficiency} \equiv \lambda_{\max}[\text{Cov}(\hat{\theta} \mid \theta_{true}, \mathbf{w}_{opt}(\theta_{true})))]/\lambda_{\max}[\text{Cov}(\hat{\theta} \mid \theta_{true}, \mathbf{w}_\rho(\theta_0))],$$

where $\mathbf{w}_{opt}(\theta)$ is the E-optimal design for parameter $\theta$. They show that for appropriately chosen values of $\rho$, the robust design is more *efficient* than the optimal design, if the initial parameter $\theta_0$ does not equal the true parameter.

While their results are very promising, an arguably more natural approach than perturbing the Jacobian would be to perturb the initial parameter estimate, around which linearization is performed. E.g., if the function $f$ describes a process, which behaves characteristically differently in different "phases", and the parameter $\theta$ controls which of the phases the process is in, then a robust design should intuitively "hedge" the design against the behavior in each possible phase. In such a case, the uniform distribution (which the robust SDP chooses for large $\rho$) would not be the most robust design.

If we discretize the space of possible parameter perturbations (within a reasonably chosen interval), we can use SATURATE to find robust experimental designs. While the classical E-optimality is not submodular [2], Bayesian A-optimality is (often) submodular [1, 2]. Here, the goal is to minimize the *trace* instead of eigenvalue size as error metric. Furthermore, we equip the parameters $\theta$ with an uninformative normal prior (which we chose as $\text{diag}([20^2, 20^2])$), and then minimize the expected trace of the posterior error covariance, $\text{tr}(\Sigma_{\theta|\mathcal{A}})$. Hereby, $\mathcal{A}$ is a discrete design of 20 experiments, where each option $x_i$ can be chosen repeatedly. In order to apply SATURATE, for each $\theta$, we define $F_\theta(\mathcal{A})$ as the normalized variance reduction $F_\theta(\mathcal{A}) = \frac{1}{Z_\theta}(\sigma_\theta^2 - \sigma_{\theta|\mathcal{A}}^2)$. The normalization $Z_\theta$ is chosen such that $F_\theta(\mathcal{A}) = 1$ if $\mathcal{A} = \text{argmax}_{|\mathcal{A}'|=20} F_\theta(\mathcal{A}')$, i.e., if $\mathcal{A}$ is chosen to maximize only $F_\theta$. SATURATE is then used to maximize the worst-case normalized variance reduction.

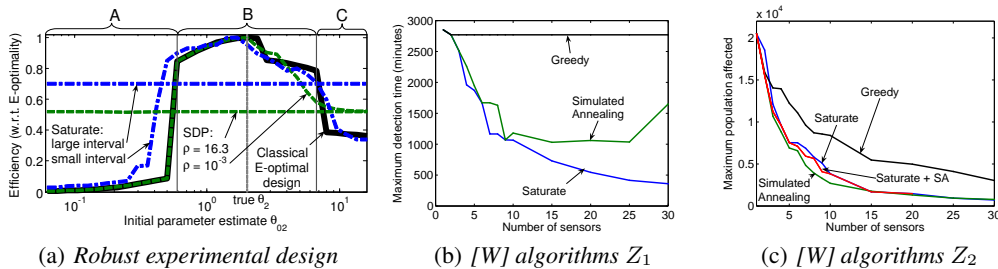

(a) *Robust experimental design*  (b) *[W] algorithms $Z_1$*  (c) *[W] algorithms $Z_2$*

Figure 2: (a) Efficiency of robust SDP of [7] and SATURATE on a biological experimental design problem. For a large range of initial parameter estimates, SATURATE outperforms the SDP solutions. (b,c) SATURATE, greedy and SA in the water network setting, when optimizing worst-case detection time ($Z_1$) and affected population ($Z_2$). SATURATE performs comparably to SA for $Z_2$ and strictly outperforms SA for $Z_1$.

We reproduced the experiment of [7], where the initial estimate of the second component $\theta_{02}$ of $\theta_0$ was varied between 0 and 16, the "true" value being $\theta_2 = 2$. For each initial estimate of $\theta_{02}$, we computed a robust design, using the SDP approach and using SATURATE, and compared them using the efficiency metric of [7]. We first optimized designs which are robust against a small perturbation of the initial parameter estimate. For the SDP, we chose a robustness parameter $\rho = 10^{-3}$, as reported in [7]. For SATURATE, we considered an interval around $[\theta\frac{1}{1+\varepsilon}, \theta(1 + \varepsilon)]$, discretized in a $5 \times 5$ grid, with $\varepsilon = .1$. Fig. 2(a) shows three characteristically different regions, $A$, $B$, $C$, separated by vertical lines. In region $B$ which contains the true parameter setting, the E-optimal design (which is optimal if the true parameter is known, i.e., $\theta_{02} = \theta_2$) performs similar to both robust methods. Hence, in region B (i.e., small deviation from the true parameter), robustness is not really necessary. Outside of region B however, where the standard E-optimal design performs badly, both robust designs do not perform well either. This is an intuitive result, as they were optimized to be robust only to small parameter perturbations.

Consequently, we compared designs which are robust against a *large* parameter range. For SDP, we chose $\rho = 16.3$, which is the maximum spectral variation of the Jacobian when we consider all initial estimates from $\theta_{02}$ varying between 0 and 16. For SATURATE, we optimized a single design which achieves the maximum normalized variance reduction over all values of $\theta_{02}$ between 0 and 16. Fig. 2(a) shows, that in this case, the design obtained by SATURATE achieves an efficiency of 69%, whereas the efficiency of the SDP design is only 52%. In the regions A and C, the SATURATE design strictly outperforms the other robust designs. This experiment indicates that designs which are robust against a large range of initial parameter estimates, as provided by SATURATE, can be more efficient than designs which are robust against perturbations of the Jacobian (the SDP approach).

**Outbreak Detection.** Consider a city water distribution network, delivering water to households via a system of pipes, pumps, and junctions. Accidental or malicious intrusions can cause contaminants to spread over the network, and we want to select a few locations (pipe junctions) to install sensors, in order to detect these contaminations as quickly as possible. In August 2006, the Battle of Water Sensor Networks (BWSN) [16] was organized as an international challenge to find the best sensor placements for a real (but anonymized) metropolitan water distribution network, consisting of 12,527 nodes. In this challenge, a set of intrusion scenarios is specified, and for each scenario a realistic simulator provided by the EPA [17] is used to simulate the spread of the contaminant for a 48 hour period. An intrusion is considered detected when one selected node shows positive contaminant concentration. BWSN considered a variety of impact measures, including the time to detection (called $Z_1$), and the size of the affected population calculated using a realistic disease model ($Z_2$). The goal of BWSN was to minimize the *expectation* of the impact measures $Z_1$ and $Z_2$ given a *uniform distribution* over intrusion scenarios.

In this paper, we consider the *adversarial* setting, where an opponent chooses the contamination scenario with knowledge of the sensor locations. The objective functions $Z_1$ and $Z_2$ are in fact submodular for a fixed intrusion scenario [9], and so the adversarial problem of minimizing the impact of the worst possible intrusion fits into our model. For these experiments, we consider scenarios which affect at least 10% of the network, resulting in a total of 3424 scenarios. Figures 2(b) and 2(c) compare the greedy algorithm, SATURATE and the simulated annealing (SA) algorithm for the problem of maximizing the worst-case detection time ($Z_1$) and worst-case affected population ($Z_2$).

Interestingly, the behavior is very different for the two objectives. For the affected population ($Z_2$), greedy performs reasonably, and SA sometimes even outperforms SATURATE. For the detection

time ($Z_1$), however, the greedy algorithm did not improve the objective at all, and SA performs poorly. The reason is that for $Z_2$, the maximum achievable scores, $F_i(\mathcal{V})$, vary drastically, since some scenarios have much higher impact than others. Hence, there is a strong "gradient", as the adversarial objective changes quickly when the high impact scenarios are covered. This gradient allows greedy and SA to work well. On the contrary, for $Z_1$, the maximum achievable scores, $F_i(\mathcal{V})$, are constant, since all scenarios have the same simulation duration. Unless *all* scenarios are detected, the worst-case detection time stays constant at the simulation length. Hence, many node exchange proposals considered by SA, as well as the addition of a new sensor location by greedy, do not change the adversarial objective, and the algorithms have no useful performance metric. Similarly to the GP Kriging setting, our results show that optimizing the worst-case score leads to reasonable performance in the average case score, but not necessarily vice versa.

## 5 Conclusions

In this paper, we considered the problem of selecting observations which are informative with respect to an objective function chosen by an adversary. We demonstrated how this class of problems encompasses the problem of finding designs which minimize the maximum posterior variance in Gaussian Processes regression, robust experimental design, and detecting events spreading over graphs. In each of these settings, the individual objectives are submodular and can be approximated well using, e.g., the greedy algorithm; the adversarial objective, however, is not submodular. We proved that there cannot exist any approximation algorithm for the adversarial problem if the constraint on the observation set size must be exactly met, unless $\mathrm{P} = \mathrm{NP}$. Consequently, we presented an efficient approximation algorithm, SATURATE, which finds observation sets which are guaranteed to be least as informative as the optimal solution, and only logarithmically more expensive. In a strong sense, this guarantee is the best possible. We extensively evaluated our algorithm on several real-world problems. For Gaussian Process regression, we showed that SATURATE compares favorably to state-of-the-art heuristics, while being simpler, faster, and providing theoretical guarantees. For robust experimental design, SATURATE performs favorably compared to SDP based approaches.

**Acknowledgements** This work was partially supported by NSF Grants No. CNS-0509383, CNS-0625518, CCF-0448095, CCF-0729022, and a gift from Intel. Anupam Gupta and Carlos Guestrin were partly supported by Alfred P. Sloan Fellowships, Carlos Guestrin by an IBM Faculty Fellowship and Andreas Krause by a Microsoft Research Graduate Fellowship.

## Footnotes

[1] This bound is only meaningful for integral $F_i$, otherwise it could be arbitrarily improved by scaling the $F_i$.

## References

[1] A. Das and D. Kempe. Algorithms for subset selection in linear regression. In *Manuscript*, 2007.

[2] A. Krause, A. Singh, and C. Guestrin. Near-optimal sensor placements in Gaussian processes: Theory, efficient algorithms and empirical studies. In *To appear in the JMLR*, 2007.

[3] G. Nemhauser, L. Wolsey, and M. Fisher. An analysis of the approximations for maximizing submodular set functions. *Mathematical Programming*, 14:265–294, 1978.

[4] U. Feige. A threshold of ln n for approximating set cover. *J. ACM*, 45(4), 1998.

[5] A. Krause, B. McMahan, C. Guestrin, and A. Gupta. Robust submodular observation selection. Technical report, CMU-ML-08-100, 2008.

[6] C. E. Rasmussen and C. K. I. Williams. *Gaussian Process for Machine Learning*. Adaptive Computation and Machine Learning. MIT Press, 2006.

[7] P. Flaherty, M. Jordan, and A. Arkin. Robust design of biological experiments. In *NIPS*, 2006.

[8] S. Boyd and L. Vandenberghe. *Convex Optimization*. Cambridge UP, March 2004.

[9] J. Leskovec, A. Krause, C. Guestrin, C. Faloutsos, J. VanBriesen, and N. Glance. Cost-effective outbreak detection in networks. In *KDD*, 2007.

[10] T. Fujito. Approximation algorithms for submodular set cover with applications. *TIEICE*, 2000.

[11] L.A. Wolsey. An analysis of the greedy algorithm for the submodular set covering problem. *Combinatorica*, 2:385–393, 1982.

[12] T. G. Robertazzi and S. C. Schwartz. An accelerated sequential algorithm for producing D-optimal designs. *SIAM Journal of Scientific and Statistical Computing*, 10(2):341–358, March 1989.

[13] M. Widmann and C. S. Bretherton. 50 km resolution daily precipitation for the pacific northwest. http://www.jisao.washington.edu/data_sets/widmann/, May 1999.

[14] J. Sacks and S. Schiller. *Statistical Decision Theory and Related Topics IV, Vol. 2*. Springer, 1988.

[15] D. P. Wiens. Robustness in spatial studies ii: minimax design. *Environmetrics*, 16:205–217, 2005.

[16] A. Ostfeld, J. G. Uber, and E. Salomons. Battle of water sensor networks: A design challenge for engineers and algorithms. In *8th Symposium on Water Distribution Systems Analysis*, 2006.

[17] L. A. Rossman. The epanet programmer's toolkit for analysis of water distribution systems. In *Annual Water Resources Planning and Management Conference*, 1999.

